# Reconstructing Stimulus-Driven Neural Networks from Spike Times

**Duane Q. Nykamp**
UCLA Mathematics Department
Los Angeles, CA 90095
nykamp@math.ucla.edu

## Abstract

We present a method to distinguish direct connections between two neurons from common input originating from other, unmeasured neurons. The distinction is computed from the spike times of the two neurons in response to a white noise stimulus. Although the method is based on a highly idealized linear-nonlinear approximation of neural response, we demonstrate via simulation that the approach can work with a more realistic, integrate-and-fire neuron model. We propose that the approach exemplified by this analysis may yield viable tools for reconstructing stimulus-driven neural networks from data gathered in neurophysiology experiments.

## 1 Introduction

The pattern of connectivity between neurons in the brain is fundamental to understanding the function the brain's neural networks. Related properties of closely connected neurons, for example, may lead to inferences on how the observed properties are built or enhanced by the neural connections. Unfortunately, the complexity of higher organisms makes obtaining combined functional and connectivity data extraordinarily difficult.

The most common tool for recording *in vivo* the activity of neurons in higher organisms is the extracellular electrode. Typically, one uses this electrode to record only the times of output spikes, or action potentials, of neurons. In such an experiment, the states of the measured neurons remain hidden. The ability to infer connectivity patterns from spike times alone would greatly expand the attainable connectivity data and provide the opportunity to better address the link between function and connectivity.

Attempts to infer connectivity from spike time data have focused on second-order statistics of the spike times of two simultaneously recorded neurons. In particular, the joint peristimulus time histogram (JPSTH) and its integral, the shuffle-corrected correlogram [1, 2, 3] have become widely used tools to analyze such data.

However, the JPSTH and correlogram cannot distinguish correlations induced by connections between the two measured neurons (direct connection correlations) from correlations induced by common connections from a third, unmeasured neuron (common input correlations). Inferences from the JPSTH or correlogram about the connections between the two measured neurons are ambiguous.

Analysis tools such as partial coherence [4] can distinguish between a direct connection and common input when one can also measure neurons inducing the common input effects. The distinction of present approach is that all other neurons are unmeasured.

We demonstrate that, by characterizing how each neuron responds to the stimulus, one may be able to distinguish between direct connection and common input correlations. In that case, one could determine if a connection existed between two neurons simply by measuring their spike times in response to a stimulus. Since the properties of the neurons would be determined by the same measurements, such an analysis would be the basis for inferring links between connectivity and function.

## 2 The model

To make the subtle distinction between direct connection correlations and common input correlations, one needs to exploit an explicit model. The model must be sufficiently simple so that all necessary model parameters can be determined from experimental measurements. For this reason, the analysis is limited to phenomenological lumped models. We present analysis based on a linear-nonlinear model of neural response to white noise.

Let the stimulus $\mathbf{X}$ be a vector of independent Gaussian random variables with zero mean and standard deviation $\sigma = 1$. $\mathbf{X}$ is a discrete approximation to temporal or spatio-temporal white noise. Let $R_p^i = 1$ if neuron $p$ spiked at the discrete time point $i$ and be zero otherwise. Let the probability of a spike from a neuron be a linear-nonlinear function of the stimulus and the previous spike times of the other neurons,

$$\Pr\big(R_p^i = 1 \big| \mathbf{X} = \mathbf{x}, \mathbf{R}_q = \mathbf{r}_q, \forall q\big) = g_p\Big(\mathbf{h}_p^i \cdot \mathbf{x} + \sum_{q \neq p}\sum_{j>0} \bar{W}_{qp}^j r_q^{i-j}\Big), \qquad (1)$$

where $\mathbf{h}_p^i$ is the linear kernel of neuron $p$ shifted $i$ units in time (normalized so that $\|\mathbf{h}_p^i\| = 1$), $g_p(\cdot)$ is its output nonlinearity (representing, for example, its spike generating mechanism), and $\bar{W}_{qp}^j$ is a connectivity term representing how a spike of neuron $q$ at a particular time modifies the response of neuron $p$ after $j$ time steps.

The network of Eq. (1) is an extension of the standard linear-nonlinear model of a single neuron. The linear-nonlinear model of a single neuron can be completely reconstructed from measured spike times in response to white noise [5]. We will demonstrate that the network of linear-nonlinear neurons can be similarly analyzed to determine the connectivity between two measured neurons.

## 3 Analysis of model

Let neurons 1 and 2 be the only two measured neurons. The spike times of all other neurons will remain unmeasured. Given further simplifying assumptions detailed below, we can isolate the connectivity terms between neurons 1 and 2 ($\bar{W}_{12}^j$ and $\bar{W}_{21}^j$). We will outline a method to determined these connectivity terms from a few statistics of the two measured spikes trains and the white noise stimulus.

### 3.1 Assumptions

The first assumption is that the coupling is sufficiently weak to justify a first order approximation in the $\bar{W}_{qp}^j$. We will neglect all quadratic and higher order terms in the $\bar{W}_{qp}^j$ with one important exception. Common input correlations are second order in the $\bar{W}_{qp}^j$ because common input requires two connections. Since our analysis must include common input to the measured neurons, we retain terms containing $\bar{W}_{p1}^j \bar{W}_{q2}^k$ with $p, q > 2$.

The second assumption is that the unmeasured neurons do not respond to essentially identical stimulus features as the measured neurons (1 & 2) or each other. We quantify similarity to stimulus features by the inner product between linear kernels, $\cos \bar{\theta}_{pq}^k = \mathbf{h}_p^{i-k} \cdot \mathbf{h}_q^i$. We require each $\cos \bar{\theta}$ to be small so that we can ignore terms of the form $\bar{W} \cos \bar{\theta}$. We allow one exception and retain $\bar{W} \cos \bar{\theta}_{21}^k$ terms so that no assumption is made on the two measured neurons.

Last, we assume the nonlinearity is an error function of the form

$$g_p(x) = \frac{1}{2} \left[ 1 + \mathrm{erf} \left( \frac{x - \bar{T}_p}{\epsilon_p \sqrt{2}} \right) \right] \tag{2}$$

with parameters $\bar{T}_p$ and $\epsilon_p$, where $\mathrm{erf}(y) = \frac{2}{\sqrt{\pi}} \int_0^y e^{-t^2} dt$.

## 3.2 Outline of method

The first step in analyzing the network response is to ignore the fact that the neurons are embedded in a neural network and analyze the spike trains of neurons 1 and 2 as though each were an isolated linear-nonlinear system. Using the procedure outlined in Ref. [5], one can determine the effective linear-nonlinear parameters from the average firing rates $(E\{R_1^i\}$ and $E\{R_2^i\})$[1] and the stimulus-spike correlations $(E\{\mathbf{X}R_1^i\}$ and $E\{\mathbf{X}R_2^i\})$.

These effective linear-nonlinear parameters clearly will not match the parameters for neurons 1 and 2 in the complete system (Eq. (1)). The network connections alter the mean rates and stimulus-spike correlations of neurons 1 and 2, changing the linear-nonlinear parameters reconstructed from these measurements. Nonetheless, these effective linear-nonlinear system parameters can be written approximately as combinations of parameters from the network in Eq. (1).

The connectivity between neurons 1 and 2 can then be determined from the correlation between their spikes $(E\{R_1^i R_2^{i-k}\}$ measured at different positive and negative delays $k$ and the correlation of their spike pairs with the stimulus $(E\{\mathbf{X}R_1^i R_2^{i-k}\})$ as follows. Given our assumptions, we obtain equations linear in $\bar{W}_{12}^j$, $\bar{W}_{21}^j$, and $\bar{W}_{p1}^j \bar{W}_{q2}^{\tilde{j}}$. For each delay $k$, we obtain three equations: one from $E\{R_1^i R_2^{i-k}\}$, one from the projection of $E\{\mathbf{X}R_1^i R_2^{i-k}\}$ onto $E\{\mathbf{X}R_1^i\}$, and one from the projection of $E\{\mathbf{X}R_1^i R_2^{i-k}\}$ onto $E\{\mathbf{X}R_2^{i-k}\}$. At first glance, it appears that the unknowns greatly outnumber the equations.

However, the system of equations is well-posed because the $\bar{W}_{p1}^j \bar{W}_{q2}^{\tilde{j}}$ appear in the same combination for each of the three equations at a given delay. In fact, we have only two sets of unknowns, which can be written as

$$\bar{W}^k = \begin{cases} \bar{W}_{12}^{-k} & \text{for } k < 0, \\ \bar{W}_{21}^k & \text{for } k > 0, \end{cases} \tag{3}$$

and

$$\bar{U}^k = \sum_{p>2} \sum_{j,\tilde{j}} c_p^{kj\tilde{j}} \bar{W}_{p1}^j \bar{W}_{p2}^{\tilde{j}}. \tag{4}$$

All other parameters in the equations were already determined in the first stage. If $N$ is the number of delays considered, then we have $3N$ linear equations and only $2N$ unknowns.

The factor $\bar{W}^k$ is the direct connection between neurons 1 and 2 (the direction of the connection depends on the sign of the delay $k$). The factor $\bar{U}^k$ is the common input to neuron 2 and neuron 1 ($k$ times steps delayed) from all other neurons in the network. The weighting

$(c_p^{kj\tilde{j}})$ of its terms depends on the properties of the unmeasured neurons. Fortunately, since we can treat $\bar{U}^k$ as a unit, we don't need to determine the weighting.

To analyze spike train data, we approximate the statistics $E\{R_1^i\}$, $E\{R_2^i\}$, $E\{\mathbf{X}R_1^i\}$, $E\{\mathbf{X}R_2^i\}$, $E\{R_1^iR_2^{i-k}\}$, and $E\{\mathbf{X}R_1^iR_2^{i-k}\}$ by averages over an experiment. We then compute the least-squares fit to solve for approximations of $\bar{W}$ and $\bar{U}$. We denote these approximations (or correlation measures) as $\mathcal{W}$ and $\mathcal{U}$, respectively.

## 4  Demonstration

We demonstrate the ability of the measures $\mathcal{W}$ and $\mathcal{U}$ to distinguish direct connection correlations from common input correlations with three example simulations. In the first two examples, we simulated a network of three coupled linear-nonlinear neurons (Eqs. (1) and (2)). In the third example, we simulated a pair of integrate-and-fire neurons driven by the stimulus in a manner similar to the linear-nonlinear neurons. In each example, we measured only the spike times of neuron 1 and neuron 2.

Since the white noise stimulus does not repeat, one cannot calculate a JPSTH or shuffle-corrected correlogram. Instead, for comparison we calculated the covariance between the spike times, $\mathcal{C}^k = \langle R_1^iR_2^{i-k}\rangle - \langle R_1^i\rangle\langle R_2^{i-k}\rangle$, and a stimulus independent correlation measure introduced in Ref. [6], $\mathcal{S}^k = \langle R_1^iR_2^{i-k}\rangle - \nu_{21}^k$, where $\langle\rangle$ represents averaging over the entire stimulus. The quantity $\nu_{21}^k$ is the expected value of $\langle R_1^iR_2^{i-k}\rangle$ if neurons 1 and 2 were independent linear-nonlinear systems responding to the same stimulus.

We used spatio-temporal linear kernels of the form

$$\mathbf{h}_p(\mathbf{j}, t) = te^{-\frac{t}{\tau_h}}e^{-\frac{|\mathbf{j}|^2}{40}}\sin((j_1\cos\phi_p + j_2\sin\phi_p)f_p + k_p) \tag{5}$$

for $t > 0$ ($\mathbf{h}_p = 0$ otherwise), where $\mathbf{j} = (j_1, j_2)$ denotes a discrete space point. For the linear-nonlinear simulations, we sampled this function on a $20 \times 20 \times 20$ grid in space and time, normalizing the resulting vector to obtain the unit vector $\mathbf{h}_p^i$. The kernels were chosen to be caricatures of receptive fields of simple cells in visual cortex. The only geometry of the kernels that appears in the equations is their inner products $\cos\bar{\theta}_{pq}^k = \mathbf{h}_p^{i-k}\cdot\mathbf{h}_q^i$.

For the first example, we simulated a network of three linear-nonlinear neurons. Neuron 2 had an excitatory connection onto neuron 1 with a delay of 5–6 units of time (a positive delay for our sign convention): $\bar{W}_{21}^5 = \bar{W}_{21}^6 = 0.6$. Neuron 3 had one excitatory connection onto neuron 1 and second excitatory connection onto neuron 2 that was delayed by 6–8 units of time (a negative delay): $\bar{W}_{31}^1 = \bar{W}_{31}^2 = \bar{W}_{32}^8 = \bar{W}_{32}^9 = 1.5$. In this way, the spike times from neuron 1 and 2 had positive correlations due to both a direct connection and common input. Fig. 1 shows the results after simulating for 600,000 units of time, obtaining 16,000–22,000 spikes per neuron.

The covariance $\mathcal{C}$ has peaks at both positive and negative delays, corresponding to the direct connection and common input, respectively, as well as a small peak around zero due to the shared stimulus (see Ref. [6]). The measure $\mathcal{S}$ eliminates the stimulus-induced correlation, but still cannot distinguish the direct connection from the common input. The proposed measures $\mathcal{W}$ and $\mathcal{U}$, however, do separate the two sources of correlation. $\mathcal{W}$ contains a peak only at the positive delay corresponding to the direct connection from neuron 2 to neuron 1; $\mathcal{U}$ contains a peak only at the negative delay corresponding to the common input from the (unmeasured) third neuron. This distinction was made at the cost of a dramatic increase in the noise. On the order of 20,000 spikes were needed to get clean results even in this idealized simulation, a long experiment given the typically low firing rates in response to white noise stimuli.

Theoretically, the method should handle inhibitory connections just as well as excitatory

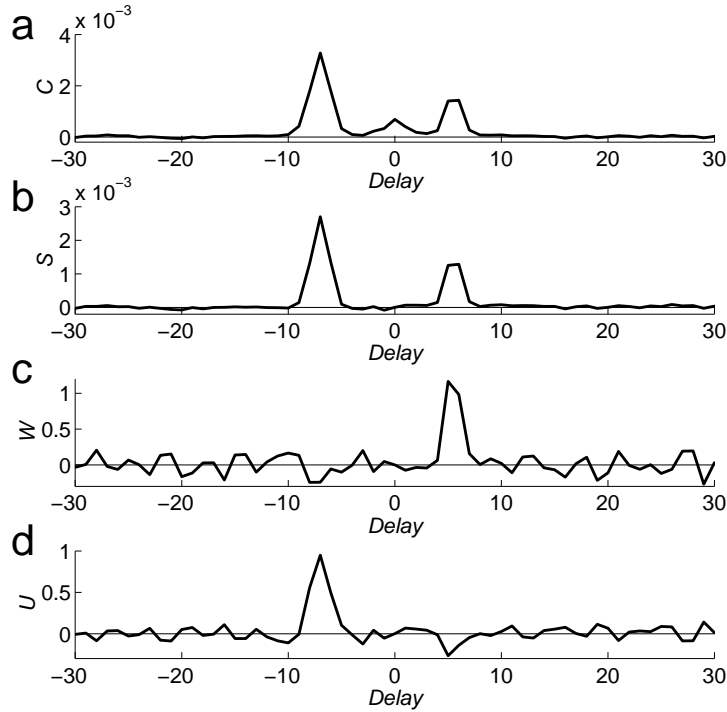

Figure 1: Results from the spike times of two neurons in a simulation of three linear-nonlinear neurons. Delay is in units of time and is the spike time of neuron 1 minus the spike time of neuron 2. The correlations at a positive delay are due to a direct connection, while those a negative delay are due to common input. **(a)** The covariance $\mathcal{C}$ between the spike times of neuron 1 and neuron 2 reflects both connections. The third peak around zero delay, due to similarity in the kernels $\mathbf{h}_1^i$ and $\mathbf{h}_2^i$, is induced by the common stimulus. **(b)** The correlation measure $\mathcal{S}$ removes the correlation induced by the common stimulus, but cannot distinguish between the two connectivity induced correlations. **(c–d)** The measures $\mathcal{W}$ and $\mathcal{U}$ do distinguish the connectivity induced correlations. $\mathcal{W}$ reflects only the direct connection (c); $\mathcal{U}$ reflects only the common input (d). Parameters for $g(\cdot)$: $\bar{T}_1 = 2.5$, $\bar{T}_2 = 3.0$, $\bar{T}_3 = 2.2$, $\epsilon_1 = 0.5$, $\epsilon_2 = 1.0$, $\epsilon_3 = 0.7$. Parameters for $\mathbf{h}$: $\tau_h = 1$, $\phi_1 = 0$, $\phi_2 = \pi/8$, $\phi_3 = \pi/4$, $f_1 = 0.5$, $f_2 = 0.8$, $f_3 = 1.0$, $k_1 = 0$, $k_2 = -1$, $k_3 = 1$.

connections. To test the inhibitory case, we modified the connections so that neuron 1 received an inhibitory connection from neuron 2 ($\bar{W}_{21}^5 = \bar{W}_{21}^6 = -0.3$), and neuron 1 received an inhibitory connection from neuron 3 ($\bar{W}_{31}^1 = \bar{W}_{31}^2 = -1.0$). Neuron 2 continued to receive an excitatory connection from neuron 3 ($\bar{W}_{32}^8 = \bar{W}_{32}^9 = 1.0$). The low firing rates of neurons, however, makes inhibition more difficult to detect via correlations [3]. Similarly, the measures $\mathcal{W}$ and $\mathcal{U}$ performed less well with inhibition. To demonstrate that they could, at least theoretically, work for inhibition, we increased the firing rates, used $\bar{W}$s with smaller magnitudes, and increased the simulation length. Fig. 2 shows the results after simulating for 1,200,000 units of time, obtaining 130,000–140,000 spikes per neuron. With this extraordinarily large number of spikes, $\mathcal{W}$ and $\mathcal{U}$ successfully distinguish the direct connection correlations from the common input correlations.

To test the robustness of the method to deviations from the linear-nonlinear model, we simulated a system of two integrate-and-fire neurons whose input was a threshold-linear function of the stimulus. The neurons received common input from a threshold-linear unit,

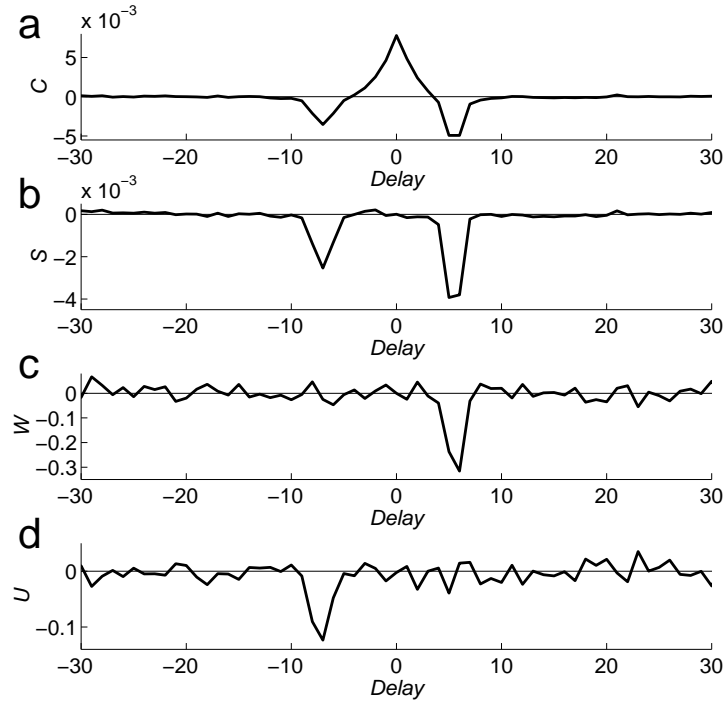

Figure 2: Results from the simulation of the same linear-nonlinear network as in Fig. 1, except that the connections from both neuron 2 and neuron 3 onto neuron 1 were made inhibitory. Panels are as in Fig. 1. Again, $\mathcal{S}$ eliminates the stimulus-induced peak in $\mathcal{C}$. $\mathcal{W}$ reflects only the direct connection correlations, and $\mathcal{U}$ reflects only the common input correlations. This inhibitory example, however, required a long simulation for accurate results (see text). Parameters for $g(\cdot)$: $\bar{T}_1 = 1.2$, $\bar{T}_2 = 2.0$, $\bar{T}_3 = 1.5$, $\epsilon_1 = 0.5$, $\epsilon_2 = 1.0$, $\epsilon_3 = 0.7$. Parameters for $\mathbf{h}$ are the same as in Fig. 1.

and neuron 1 received a direct connection from neuron 2 (see Fig. 3).

We let $t$ be given in milliseconds, sampled Eq. (5) on a $20 \times 20 \times 30$ grid in space and time, using a 2 ms grid in time, and normalized the resulting vector to obtain the unit vector $\mathbf{h}_p^i$. A two millisecond sample rate of discrete white noise is unrealistic in many experiments, so we departed further from the assumptions of the derivation and let the stimulus be white noise sampled at 10 ms. We let the stimulus standard deviation be $\sigma = 1/\sqrt{5}$ so that it had the same power as discrete white noise sampled at 2 ms with $\sigma = 1$.

After one hour of simulated time (360,000 frames), we collected approximately 23,000–25,000 spikes per neuron. Fig. 4 shows that the method still effectively distinguishes direct connection correlations from common input correlations. The separation isn't perfect as $\mathcal{W}$ becomes negative where the common input correlation is positive and $\mathcal{U}$ becomes negative where the direct input correlation is positive. To determine whether a combination of positive $\mathcal{W}$ and negative $\mathcal{U}$, for example, indicates positive direct connection correlation or negative common input correlation, one simply needs to look to see if $\mathcal{S}$ is positive or negative.

Fig. 4 dramatically illustrates the increased noise in $\mathcal{W}$ and $\mathcal{U}$. For this reason, the measures are useful only when one can run a relatively long experiment to get an acceptable signal-to-noise ratio. The noise is due to the conditioning of the (non-square) matrix in the

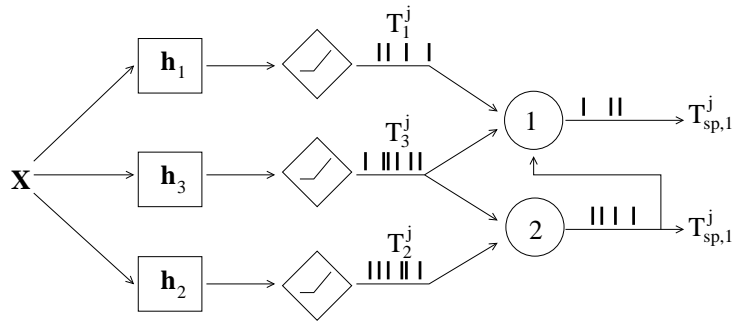

Figure 3: Diagram of two integrate-and-fire neurons (circles) receiving threshold-linear input from the stimulus. The neurons received common input from threshold-linear unit 3, and neuron 1 received a direct connection from neuron 2. The evolution of the voltage of neuron $p$ in response to input $g_p(t)$ was given by $\tau_m \frac{dV_p}{dt} + V_p + g_p(t)(V_p - \mathcal{E}_s) = 0$. When $V_p(t)$ reached 1, a spike was recorded, and the voltage was reset to 0 and held there for an absolute refractory period of length $\tau_{ref}$. We let $g_p(t) = g_p^{ext}(t) + g_p^{int}(t)$, where the external input was $g_p^{ext}(t) = 0.05 \sum_j G(t - T_p^j) + 0.05 \sum_j G(t - T_3^j - \delta_p)$ with $G(t) = \frac{e^2}{4}\left(\frac{t}{\tau_s}\right)^2 e^{-t/\tau_s}$ for $t > 0$ and $G(t) = 0$ otherwise. The $T_p^j$ were drawn from a modulated Poisson process with rate given by $\alpha_p \left[\mathbf{h}_p^i \cdot \mathbf{X}\right]^+$ where $[x]^+ = x$ if $x > 0$ and is zero otherwise. The internal input $g_2^{int}(t)$ to neuron 2 was set to zero, and the internal input to neuron 1 was set to reflect an excitatory connection from neuron 2, $g_1^{int}(t) = 0.05 \sum_j G(t - T_{sp,2}^j - \delta_{21})$, where the $T_{sp,2}^j$ are the spike times of neuron 2.

least-square calculation of $\mathcal{W}$ and $\mathcal{U}$. The condition numbers in the three examples were approximately 70, 50, and 110, respectively. Measurement errors or noise could be magnified by as much as these factors. The high condition numbers reflect the subtlety of the distinction we are making.

Obtaining values of $\mathcal{W}$ and $\mathcal{U}$ significantly beyond the noise level in real experiments may prove a formidable challenge. However, the utility of $\mathcal{W}$ and $\mathcal{U}$ with noisy data greatly improves when they are used in conjunction with other measures. One can use a less noisy measure such as $\mathcal{S}$ to find significant stimulus-independent correlations and determine their magnitudes. Then, assuming one can rule out causes like covariation in latency or excitability [7], one simply needs to determine if the correlations were caused by a direct connection or by common input. One does not need to use $\mathcal{W}$ and $\mathcal{U}$ to reject the null hypothesis of no connectivity-induced correlations; they are needed only to make the remaining binary distinction.

The proposed method should be viewed simply as an example of a new framework for reconstructing stimulus-driven neural networks. Clearly, extensions beyond the presented model will be necessary since the linear-nonlinear model can adequately describe the behavior of only a small subset of neurons in primary sensory areas. Furthermore, methods to validate the assumed model will be required before results of this approach can be trusted.

Though limited in scope and model-dependent, we have demonstrated what appears to be the first example of a definitive dissociation between direct connection and common input correlations from spike time data. At least in the case of excitatory connections, this distinction can be made with a realistic, albeit large, amount of data. With further refinements, this approach may yield viable tools for reconstructing stimulus-driven neural networks.

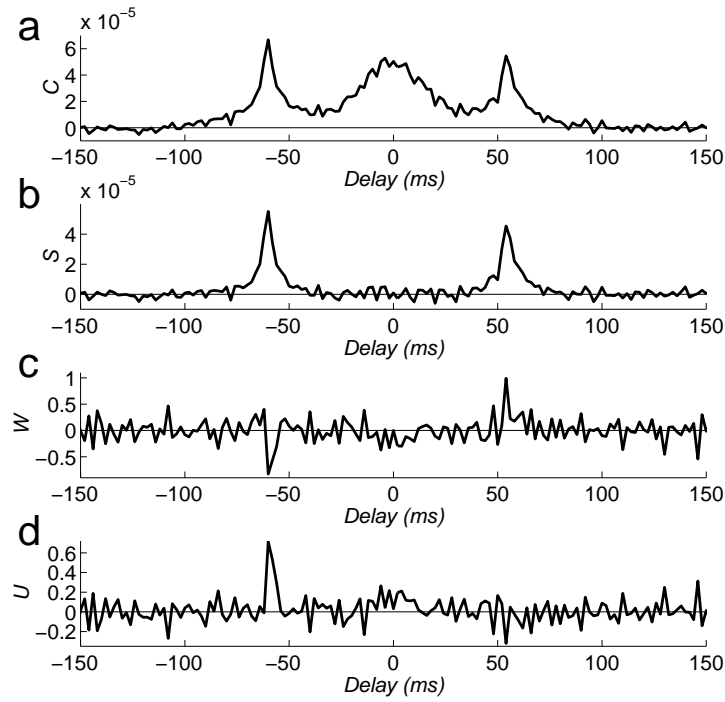

Figure 4: Results from the simulation of two integrate-and-fire neurons, where neuron 2 had an excitatory connection onto neuron 1 with a delay $\delta_{21} = 50$ ms. Both neurons received common input, but the common input to neuron 2 was delayed ($\delta_1 = 0$ ms, $\delta_2 = 60$ ms). Panels are as in Fig. 1. $\mathcal{S}$ greatly reduces the central, stimulus-induced correlation from $\mathcal{C}$. $\mathcal{W}$ and $\mathcal{U}$ successfully distinguish the direct connection correlations from the common input correlations, but also negatively reflect each other. Ambiguity in interpretation of $\mathcal{W}$ and $\mathcal{U}$ can be eliminated by comparison with $\mathcal{S}$. Integrate-and-fire parameters: $\tau_m = 5$ ms, $\mathcal{E}_s = 6.5$, $\tau_2 = 2$ ms, $\tau_{ref} = 2$ ms, $\alpha_1 = \alpha_2 = 0.25$ ms$^{-1}$, and $\alpha_3 = 0.1$ ms$^{-1}$. Parameters for **h** are the same as in Fig. 1 except that $\tau_h = 10$ ms.

## Footnotes

[1] $E\{\cdot\}$ denotes expected value.

## References

[1] D. H. Perkel, G. L. Gerstein, and G. P. Moore. Neuronal spike trains and stochastic point processes. II. Simultaneous spike trains. *Biophys. J.*, 7:419–40, 1967.

[2] A. M. H. J. Aertsen, G. L. Gerstein, M. K. Habib, and G. Palm. Dynamics of neuronal firing correlation: Modulation of "effective connectivity". *J. Neurophysiol.*, 61:900–917, 1989.

[3] G. Palm, A. M. H. J. Aertsen, and G. L. Gerstein. On the significance of correlations among neuronal spike trains. *Biol. Cybern.*, 59:1–11, 1988.

[4] J. R. Rosenberg, A. M. Amjad, P. Breeze, D. R. Brillinger, and D. M. Halliday. The Fourier approach to the identification of functional coupling between neuronal spike trains. *Prog. Biophys. Mol. Biol.*, 53:1–31, 1989.

[5] D. Q. Nykamp and Dario L. Ringach. Full identification of a linear-nonlinear system via cross-correlation analysis. *J. Vision*, 2:1–11, 2002.

[6] D. Q. Nykamp. A spike correlation measure that eliminates stimulus effects in response to white noise. *J. Comp. Neurosci.*, 14:193–209, 2003.

[7] C. D. Brody. Correlations without synchrony. *Neural. Comput.*, 11:1537–51, 1999.
